# Graph-based Consensus Maximization among Multiple Supervised and Unsupervised Models

**Jing Gao[†], Feng Liang[†], Wei Fan[‡], Yizhou Sun[†], and Jiawei Han[†]**
[†]University of Illinois at Urbana-Champaign, IL USA
[‡]IBM TJ Watson Research Center, Hawthorn, NY USA
[†]{jinggao3,liangf,sun22,hanj}@illinois.edu, [‡]weifan@us.ibm.com

## Abstract

Ensemble classifiers such as bagging, boosting and model averaging are known to have improved accuracy and robustness over a single model. Their potential, however, is limited in applications which have no access to raw data but to the meta-level model output. In this paper, we study ensemble learning with output from multiple supervised and unsupervised models, a topic where little work has been done. Although unsupervised models, such as clustering, do not directly generate label prediction for each individual, they provide useful constraints for the joint prediction of a set of related objects. We propose to consolidate a classification solution by maximizing the consensus among both supervised predictions and unsupervised constraints. We cast this ensemble task as an optimization problem on a bipartite graph, where the objective function favors the smoothness of the prediction over the graph, as well as penalizing deviations from the initial labeling provided by supervised models. We solve this problem through iterative propagation of probability estimates among neighboring nodes. Our method can also be interpreted as conducting a constrained embedding in a transformed space, or a ranking on the graph. Experimental results on three real applications demonstrate the benefits of the proposed method over existing alternatives[1].

## 1   Introduction

We seek to integrate knowledge from multiple information sources. Traditional ensemble methods such as bagging, boosting and model averaging are known to have improved accuracy and robustness over a single model. Their potential, however, is limited in applications which have no access to raw data but to the meta-level model output. For example, due to privacy, companies or agencies may not be willing to share their raw data but their final models. So information fusion needs to be conducted at the decision level. Furthermore, different data sources may have different formats, for example, web video classification based on image, audio and text features. In these scenarios, we have to combine incompatible information sources at the coarser level (predicted class labels) rather than learn the joint model from raw data.

In this paper, we consider the general problem of combining *output* of multiple supervised and unsupervised models to improve prediction accuracy. Although unsupervised models, such as clustering, do not directly generate label predictions, they provide useful constraints for the classification task. The rationale is that objects that are in the same cluster should be more likely to receive the same class label than the ones in different clusters. Furthermore, incorporating the unsupervised clustering models into classification ensembles improves the base model diversity, and thus has the potential of improving prediction accuracy.

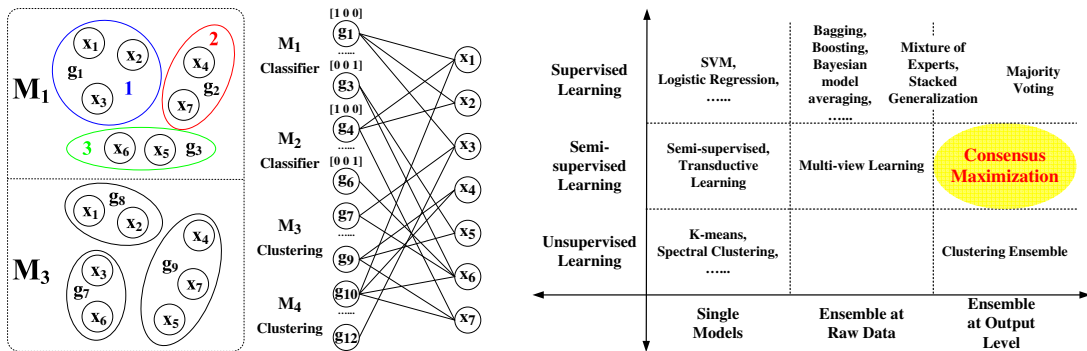

Figure 1: Groups      Figure 2: Bipartite Graph   Figure 3: Position of Consensus Maximization

Suppose we have a set of data points $X = \{x_1, x_2, \ldots, x_n\}$ from $c$ classes. There are $m$ models that provide information about the classification of $X$, where the first $r$ of them are (supervised) classifiers, and the remaining are (unsupervised) clustering algorithms. Consider an example where $X = \{x_1, \ldots, x_7\}$, $c = 3$ and $m = 4$. The output of the four models are:

$$M_1 = \{1,1,1,2,3,3,2\} \quad M_2 = \{1,1,2,2,2,3,1\} \quad M_3 = \{2,2,1,3,3,1,3\} \quad M_4 = \{1,2,3,1,2,1,1\}$$

where $M_1$ and $M_2$ assign each object a class label, whereas $M_3$ and $M_4$ simply partition the objects into three clusters and assign each object a cluster ID. Each model, no matter it is supervised or unsupervised, partitions $X$ into groups, and objects in the same group share either the same predicted class label or the same cluster ID. We summarize the data, models and the corresponding output by a bipartite graph. In the graph, nodes at the left denote the groups output by the $m$ models with some labeled ones from the supervised models, nodes at the right denote the $n$ objects, and a group and an object are connected if the object is assigned to the group by one of the models. For the aforementioned toy example, we show the groups obtained from a classifier $M_1$ and a clustering model $M_3$ in Figure 1, as well as the group-object bipartite graph in Figure 2.

The objective is to predict the class label of $x_i \in X$, which agrees with the base classifiers' predictions, and meanwhile, satisfies the constraints enforced by the clustering models, as much as possible. To reach maximum consensus among all the models, we define an optimization problem over the bipartite graph whose objective function penalizes deviations from the base classifiers' predictions, and discrepancies of predicted class labels among nearby nodes. In the toy example, the consensus label predictions for $X$ should be $\{1,1,1,2,2,3,2\}$.

**Related Work.** We summarize various learning problems in Figure 3, where one dimension represents the goal – from unsupervised to supervised, and the other dimension represents the method – single models, ensembles at the raw data, or ensembles at the output level. Our proposed method is a semi-supervised ensemble working at the output level, where little work has been done.

Many efforts have been devoted to develop single-model learning algorithms, such as Support Vector Machines and logistic regression for classification, K-means and spectral clustering for clustering. Recent studies reveal that unsupervised information can also be utilized to improve the accuracy of supervised learning, which leads to semi-supervised [29, 8] and transductive learning [21]. Although our proposed algorithm works in a transductive setting, existing semi-supervised and transductive learning methods cannot be easily applied to our problem setting and we discuss this in more detail at the end of Section 2. Note that all methods listed in Figure 3 are for single task learning. On the contrary, multi-task learning [6, 9] deals with multiple tasks simultaneously by exploiting dependence among tasks, which has a different problem setting and thus is not discussed here.

In Figure 3, we divide ensemble methods into two categories depending on whether they require access to raw data. In unsupervised learning, many clustering ensemble methods [12, 17, 25, 26] have been developed to find a consensus clustering from multiple partitionings without accessing the features. In supervised learning, however, only majority voting type algorithms work on the model output level, and most well-known classification ensemble approaches [2, 11, 19] (eg. bagging, boosting, bayesian model averaging) involve training diversified classifiers from raw data. Methods such as mixture of experts [20] and stacked generalization [27] try to obtain a meta-learner on top of the model output, however, they still need the labels of the raw data as feedbacks, so we position them as an intermediate between raw data ensemble and output ensemble. In multi-view

learning [4, 13], a joint model is learnt from both labeled and unlabeled data from multiple sources. Therefore, it can be regarded as a semi-supervised ensemble requiring access to the raw data.

**Summary.** The proposed consensus maximization problem is a challenging problem that cannot be solved by simple majority voting. To achieve maximum agreement among various models, we must seek a global optimal prediction for the target objects. In Section 2, we formally define the graph-based consensus maximization problem and propose an iterative algorithm to solve it. The proposed solution propagates labeled information among neighboring nodes until stabilization. We also present two different interpretations of the proposed method in Section 3, and discuss how to incorporate feedbacks obtained from a few labeled target objects into the framework in Section 4. An extensive experimental study is carried out in Section 5, where the benefits of the proposed approach are illustrated on 20 Newsgroup, Cora research papers, and DBLP publication data sets.

## 2 Methodology

Suppose we have the output of $r$ classification algorithms and $(m - r)$ clustering algorithms on a data set $X$. For the sake of simplicity, we assume that each point is assigned to only one class or cluster in each of the $m$ algorithms, and the number of clusters in each clustering algorithm is $c$, same as the number of classes. Note that cluster ID $z$ may not be related to class $z$. So each base algorithm partitions $X$ into $c$ groups and there are totally $v = mc$ groups, where the first $s = rc$ groups are generated by classifiers and the remaining $v - s$ groups are from clustering algorithms. Before proceeding further, we introduce some notations that will be used in the following discussion: $B_{n \times m}$ denotes an $n \times m$ matrix with $b_{ij}$ representing the $(ij)$-th entry, and $\vec{b}_{i\cdot}$ and $\vec{b}_{\cdot j}$ denote vectors of row $i$ and column $j$, respectively. See Table 1 for a summary of important symbols.

We represent the objects and groups in a bipartite graph as shown in Figure 2, where the object nodes $x_1, \ldots, x_n$ are on the right, the group nodes $g_1, \ldots, g_v$ are on the left. The affinity matrix $A_{n \times v}$ of this graph summarizes the output of $m$ algorithms on $X$:

$$a_{ij} = 1, \quad \text{if } x_i \text{ is assigned to group } g_j \text{ by one of the algorithms;} \quad 0, \quad \text{otherwise.}$$

We aim at estimating the conditional probability of each object node $x_i$ belonging to $c$ classes. As a nuisance parameter, the conditional probabilities at each group node $g_j$ are also estimated. These conditional probabilities are denoted by $U_{n \times c}$ for object nodes and $Q_{v \times c}$ for group nodes:

$$u_{iz} = \hat{P}(y = z | x_i) \quad \text{and} \quad q_{jz} = \hat{P}(y = z | g_j).$$

Since the first $s = rc$ groups are obtained from supervised learning models, they have some initial class label estimates denoted by $Y_{v \times c}$ where

$$y_{jz} = 1, \quad \text{if } g_j \text{'s predicted label is } z, j = 1, \ldots, s; \quad 0, \quad \text{otherwise.}$$

Let $k_j = \sum_{z=1}^{c} y_{jz}$, and we formulate the consensus agreement as the following optimization problem on the graph:

$$\min_{Q,U} f(Q, U) = \min_{Q,U} \left( \sum_{i=1}^{n} \sum_{j=1}^{v} a_{ij} ||\vec{u}_{i\cdot} - \vec{q}_{j\cdot}||^2 + \alpha \sum_{j=1}^{v} k_j ||\vec{q}_{j\cdot} - \vec{y}_{j\cdot}||^2 \right) \tag{1}$$

$$\text{s.t. } \vec{u}_{i\cdot} \geq \vec{0}, |\vec{u}_{i\cdot}| = 1, i = 1 : n \qquad \vec{q}_{j\cdot} \geq \vec{0}, |\vec{q}_{j\cdot}| = 1, j = 1 : v$$

where $||.||$ and $|.|$ denote a vector's L2 and L1 norm respectively. The first term ensures that if an object $x_i$ is assigned to group $g_j$ by one of the algorithm, their conditional probability estimates must be close. When $j = 1, \ldots, s$, the group node $g_j$ is from a classifier, so $k_j = 1$ and the second term puts the constraints that a group $g_j$'s consensus class label estimate should not deviate much from its initial class label prediction. $\alpha$ is the shadow price payment for violating the constraints. When $j = s + 1, \ldots, v$, $g_j$ is a group from an unsupervised model with no such constraints, and thus $k_j = 0$ and the weight of the constraint is 0. Finally, $\vec{u}_{i\cdot}$ and $\vec{q}_{j\cdot}$ are probability vectors, and therefore each component must be greater than or equal to 0 and the sum equals to 1.

We propose to solve this problem using block coordinate descent methods as shown in Algorithm 1. At the $t$-th iteration, if we fix the value of $U$, the objective function is a summation of $v$ quadratic components with respect to $\vec{q}_{j\cdot}$. The corresponding Hessian matrix is diagonal with entries equal to

**Algorithm 1** BGCM algorithm

**Input:** group-object affinity matrix $A$, initial labeling matrix $Y$; parameters $\alpha$ and $\epsilon$;
**Output:** consensus matrix $U$;
**Algorithm:**

   Initialize $U^0$,$U^1$ randomly
   $t \leftarrow 1$
   **while** $||U^t - U^{t-1}|| > \epsilon$ **do**
     $Q^t = (D_v + \alpha K_v)^{-1}(A^T U^{t-1} + \alpha K_v Y)$
     $U^t = D_n^{-1} A Q^t$
   **return** $U^t$

Table 1: Important Notations

| Symbol | Definition |
|---|---|
| $1, \ldots, c$ | class indexes |
| $1, \ldots, n$ | object indexes |
| $1, \ldots, s$ | indexes of groups from supervised models |
| $s+1, \ldots, v$ | indexes of groups from unsupervised models |
| $A_{n \times v} = [a_{ij}]$ | $a_{ij}$-indicator of object $i$ in group $j$ |
| $U_{n \times c} = [u_{iz}]$ | $u_{iz}$-probability of object $i$ wrt class $z$ |
| $Q_{v \times c} = [q_{jz}]$ | $q_{jz}$-probability of group $j$ wrt class $z$ |
| $Y_{v \times c} = [y_{jz}]$ | $y_{jz}$-indicator of group $j$ predicted as class $z$ |

$\sum_{i=1}^{n} a_{ij} + \alpha k_j > 0$. Therefore it is strictly convex and $\nabla_{\vec{q}_{j\cdot}} f(Q, U^{(t-1)}) = 0$ gives the unique global minimum of the cost function with respect to $\vec{q}_{j\cdot}$ in Eq. (2). Similarly, fixing $Q$, the unique global minimum with respect to $\vec{u}_{i\cdot}$ is also obtained.

$$\vec{q}_{j\cdot}^{(t)} = \frac{\sum_{i=1}^{n} a_{ij} \vec{u}_{i\cdot}^{(t-1)} + \alpha k_j \vec{y}_{j\cdot}}{\sum_{i=1}^{n} a_{ij} + \alpha k_j} \qquad \vec{u}_{i\cdot}^{(t)} = \frac{\sum_{j=1}^{v} a_{ij} \vec{q}_{j\cdot}^{(t)}}{\sum_{j=1}^{v} a_{ij}} \qquad (2)$$

The update formula in matrix forms are given in Algorithm 1. $D_v = \text{diag}\{(\sum_{i=1}^{n} a_{ij})\}_{v \times v}$ and $D_n = \text{diag}\{(\sum_{j=1}^{v} a_{ij})\}_{n \times n}$ act as the normalization factors. $K_v = \text{diag}\{(\sum_{z=1}^{c} y_{jz})\}_{v \times v}$ indicates the existence of constraints on the group nodes. During each iteration, the probability estimate at each group node (i.e., $Q$) receives the information from its neighboring object nodes while retains its initial value $Y$, and in return the updated probability estimates at group nodes propagate the information back to its neighboring object nodes when updating $U$. It is straightforward to prove that $(Q^{(t)}, U^{(t)})$ converges to a stationary point of the optimization problem [3].

In [14], we proposed a heuristic method to combine heterogeneous information sources. In this paper, we bring up the concept of consensus maximization and solve the problem over a bipartite graph representation. Our proposed method is related to graph-based semi-supervised learning (SSL). But existing SSL algorithms only take one supervised source (i.e., the labeled objects) and one unsupervised source (i.e., the similarity graph) [29, 8], and thus cannot be applied to combine multiple models. Some SSL methods [16] can incorporate results from an external classifier into the graph, but obviously they cannot handle multiple classifiers and multiple unsupervised sources. To apply SSL algorithms on our problem, we must first fuse all supervised models into one by some ensemble approach, and fuse all unsupervised models into one by defining a similarity function. Such a compression may lead to information loss, whereas the proposed method retains all the information and thus consensus can be reached among all the based model output.

## 3 Interpretations

In this part, we explain the proposed method from two independent perspectives.

**Constrained Embedding.** Now we focus on the "hard" consensus solution, i.e., each point is assigned to exactly one class. So $U$ and $Q$ are indicator matrices: $u_{iz} = 1$ if the ensemble assigns $x_i$ to class $z$, and 0 otherwise; similar for $q_{jz}$'s. For group nodes from classification algorithms, we will treat their entries in $Q$ as known since they have been assigned a class label by one of the classifiers, that is, $q_{jz} = y_{jz}$ for $1 \leq j \leq s$.

Because $U$ represents the consensus, we should let group $g_j$ correspond to class $z$ if majority of the objects in group $g_j$ correspond to class $z$ in the consensus solution. The optimization is thus:

$$\min_{Q,U} \quad \sum_{j=1}^{v} \sum_{z=1}^{c} \left| q_{jz} - \frac{\sum_{i=1}^{n} a_{ij} u_{iz}}{\sum_{i=1}^{n} a_{ij}} \right| \qquad (3)$$

s.t. $\sum_{z=1}^{c} u_{iz} = 1 \, \forall i \in \{1, \ldots, n\} \quad \sum_{z=1}^{c} q_{jz} = 1 \, \forall j \in \{s+1, \ldots, v\} \quad u_{iz} \in \{0, 1\} \quad q_{jz} \in \{0, 1\}$  (4)

   $q_{jz} = 1 \, \forall j \in \{1, \ldots, s\}$ if $g_j$'s label is $z$   $q_{jz} = 0 \, \forall j \in \{1, \ldots, s\}$ if $g_j$'s label is not $z$   (5)

Here, the two indicator matrices $U$ and $Q$ can be viewed as embedding $x_1, \ldots, x_n$ (object nodes) and $g_1, \ldots, g_v$ (group nodes) into a $c$-dimensional cube. Due to the constraints in Eq. (4), $\vec{u}_{i\cdot}$ and $\vec{q}_{j\cdot}$ reside on the boundary of the $(c-1)$-dimensional hyperplane in the cube. $\vec{a}_{\cdot j}$ denotes the objects group $g_j$ contains, $\vec{q}_{j\cdot}$ can be regarded as the group representative in this new space, and thus it should be close to the group mean: $\frac{\sum_{i=1}^{n} a_{ij}\vec{u}_{i\cdot}}{\sum_{i=1}^{n} a_{ij}}$. For the $s$ groups obtained from classification algorithms, we know their "ideal" embedding, as represented in the constraints in Eq. (5).

We now relate this problem to the optimization framework discussed in Section 2. $a_{ij}$ can only take value of 0 or 1, and thus Eq. (3) just depends on the cases when $a_{ij} = 1$. When $a_{ij} = 1$, no matter $q_{jz}$ is 1 or 0, we have $|q_{jz} \sum_{i=1}^{n} a_{ij} - \sum_{i=1}^{n} a_{ij} u_{iz}| = \sum_{i=1}^{n} |a_{ij}(q_{jz} - u_{iz})|$. Therefore,

$$\sum_{j:a_{ij}=1} \sum_{z=1}^{c} \left| q_{jz} - \frac{\sum_{i=1}^{n} a_{ij} u_{iz}}{\sum_{i=1}^{n} a_{ij}} \right| = \sum_{j:a_{ij}=1} \sum_{z=1}^{c} \frac{\left| q_{jz} \sum_{i=1}^{n} a_{ij} - \sum_{i=1}^{n} a_{ij} u_{iz} \right|}{\sum_{i=1}^{n} a_{ij}} = \sum_{j:a_{ij}=1} \sum_{z=1}^{c} \frac{\sum_{i=1}^{n} |a_{ij}(q_{jz} - u_{iz})|}{\sum_{i=1}^{n} a_{ij}}$$

Suppose the groups found by the base models have balanced size, i.e., $\sum_{i=1}^{n} a_{ij} = \gamma$ where $\gamma$ is a constant for $\forall j$. Then the objective function can be approximated as:

$$\sum_{j:a_{ij}=1} \sum_{z=1}^{c} \sum_{i=1}^{n} |a_{ij}(q_{jz} - u_{iz})| = \sum_{i=1}^{n} \sum_{j:a_{ij}=1} a_{ij} \sum_{z=1}^{c} |q_{jz} - u_{iz}| = \sum_{i=1}^{n} \sum_{j=1}^{v} a_{ij} \sum_{z=1}^{c} |q_{jz} - u_{iz}|$$

Therefore, when the classification and clustering algorithms generate balanced groups, with the same set of constraints in Eq. (4) and Eq. (5), the constrained embedding problem in Eq. (3) is equivalent to: $\min_{Q,U} \sum_{i=1}^{n} \sum_{j=1}^{v} a_{ij} \sum_{z=1}^{c} |q_{jz} - u_{iz}|$. It is obvious that this is the same as the optimization problem we propose in Section 2 with two relaxations: 1) We transform hard constraints in Eq. (5) to soft constraints where the ideal embedding is expressed in the initial labeling matrix $Y$ and the price for violating the constraints is set to $\alpha$. 2) $u_{iz}$ and $q_{jz}$ are relaxed to have values between 0 and 1, instead of either 0 or 1, and quadratic cost functions replace the L1 norms. So they are probability estimates rather than class membership indicators, and we can embed them anywhere on the plane.

Though with these relaxations, we build connections between the constrained embedding framework as discussed in this section with the one proposed in Section 2. Therefore, we can view our proposed method as embedding both object nodes and group nodes into a hyperlane so that object nodes are close to the group nodes they link to. The constraints are put on the group nodes from supervised models to penalize the embedding that are far from the "ideal" ones.

**Ranking on Consensus Structure.** Our method can also be viewed as conducting ranking with respect to each class on the bipartite graph, where group nodes from supervised models act as queries. Suppose we wish to know the probability of any group $g_j$ belonging to class 1, which can be regarded as the relevance score of $g_j$ with respect to example queries from class 1. Let $w_j = \sum_{i=1}^{n} a_{ij}$. In Algorithm 1, the relevance scores of all the groups are learnt using the following equation:

$$\vec{q}_{\cdot 1} = (D_v + \alpha K_v)^{-1}(A^T D_n^{-1} A \vec{q}_{\cdot 1} + \alpha K_v \vec{y}_{\cdot 1}) = D_\lambda (D_v^{-1} A^T D_n^{-1} A) \vec{q}_{\cdot 1} + D_{1-\lambda} \vec{y}_{\cdot 1}$$

where the $v \times v$ diagonal matrices $D_\lambda$ and $D_{1-\lambda}$ have $(j, j)$ entries as $\frac{w_j}{w_j + \alpha k_j}$ and $\frac{\alpha k_j}{w_j + \alpha k_j}$.

Consider collapsing the original bipartite graph into a graph with group nodes only, then $A^T A$ is its affinity matrix. After normalizing it to be a probability matrix, we have $p_{ij}$ in $P = D_v^{-1} A^T D_n^{-1} A$ represent the probability of jumping to node $j$ from node $i$. The groups that are predicted to be in class 1 by one of the supervised models have 1 at the corresponding entries in $\vec{y}_{\cdot 1}$, therefore these group nodes are "queries" and we wish to rank the group nodes according to their relevance to them.

Comparing our ranking model with PageRank model [24], there are the following relationships: 1) In PageRank, a uniform vector with entries all equal to 1 replaces $\vec{y}_{\cdot 1}$. In our model, we use $\vec{y}_{\cdot 1}$ to show our preference towards the query nodes, so the resulting scores would be biased to reflect the relevance regarding class 1. 2) In PageRank, the weights $D_\lambda$ and $D_{1-\lambda}$ are fixed constants $\lambda$ and $1 - \lambda$, whereas in our model $D_\lambda$ and $D_{1-\lambda}$ give personalized damping factors, where each group has a damping factor $\lambda_j = \frac{w_j}{w_j + \alpha k_j}$. 3) In PageRank, the value of link-votes are normalized by the number of outlinks at each node, whereas our ranking model does not normalize $p_{ij}$ on its outlinks, and thus can be viewed as an un-normalized version of personalized PageRank [18, 28]. When each base model generates balanced groups, both $\lambda_j$ and outlinks at each node become constants, and the proposed method simulates the standard personalized PageRank.

Table 2: Data Sets Description

| Data | ID | Category Labels | #target | #labeled |
|---|---|---|---|---|
| Newsgroup | 1 | comp.graphics comp.os.ms-windows.misc sci.crypt sci.electronics | 1408 | 160 |
| | 2 | rec.autos rec.motorcycles rec.sport.baseball rec.sport.hockey | 1428 | 160 |
| | 3 | sci.cypt sci.electronics sci.med sci.space | 1413 | 160 |
| | 4 | misc.forsale rec.autos rec.motorcycles talk.politics.misc | 1324 | 160 |
| | 5 | rec.sport.baseball rec.sport.hockey sci.crypt sci.electronics | 1424 | 160 |
| | 6 | alt.atheism rec.sport.baseball rec.sport.hockey soc.religion.christian | 1352 | 160 |
| Cora | 1 | Operating_Systems Programming Data_Structures_Algorithms_and_Theory | 603 | 60 |
| | 2 | Databases Hardware_and_Architecture Networking Human_Computer_Interaction | 897 | 80 |
| | 3 | Distributed Memory_Management Agents Vision_and_Pattern_Recognition | 1368 | 100 |
| | 4 | Graphics_and_Virtual_Reality Object_Oriented Planning Robotics Compiler_Design Software_Development | 875 | 100 |
| DBLP | 1 | Databases Data_Mining Machine_Learning Information_Retrieval | 3836 | 400 |

The relevance scores with respect to class 1 for group and object nodes will converge to

$$\vec{q}_{\cdot 1} = (I_v - D_\lambda D_v^{-1} A^T D_n^{-1} A)^{-1} D_{1-\lambda} \vec{y}_{\cdot 1} \quad \vec{u}_{\cdot 1} = (I_n - D_n^{-1} A D_\lambda D_v^{-1} A^T)^{-1} D_n^{-1} A D_{1-\lambda} \vec{y}_{\cdot 1}$$

respectively. $I_v$ and $I_n$ are identity matrices with size $v \times v$ and $n \times n$. The above arguments hold for the other classes as well, and thus each column in $U$ and $Q$ represents the ranking of the nodes with respect to each class. Because each row sums up to 1, they are conditional probability estimates of the nodes belonging to one of the classes.

## 4 Incorporating Labeled Information

Thus far, we propose to combine the output of supervised and unsupervised models by consensus. When the true labels of the objects are unknown, this is a reliable approach. However, incorporating labels from even a small portion of the objects may greatly refine the final hypothesis. We assume that labels of the first $l$ objects are known, which is encoded in an $n \times c$ matrix $F$:

$$f_{iz} = 1, \quad x_i\text{'s observed label is } z, i = 1, \ldots, l; \quad 0, \quad \text{otherwise.}$$

We modify the objection function in Eq. (1) to penalize the deviation of $\vec{u}_{i\cdot}$ of labeled objects from the observed label:

$$f(Q, U) = \sum_{i=1}^{n} \sum_{j=1}^{v} a_{ij} ||\vec{u}_{i\cdot} - \vec{q}_{j\cdot}||^2 + \alpha \sum_{j=1}^{v} k_j ||\vec{q}_{j\cdot} - \vec{y}_{j\cdot}||^2 + \beta \sum_{i=1}^{n} h_i ||\vec{u}_{i\cdot} - \vec{f}_{i\cdot}||^2 \quad (6)$$

where $h_i = \sum_{z=1}^{c} f_{iz}$. When $i = 1, \ldots, l$, $h_i = 1$, so we enforce the constraints that an object $x_i$'s consensus class label estimate should be close to its observed label with a shadow price $\beta$. When $i = l + 1, \ldots, n$, $x_i$ is unlabeled. Therefore, $h_i = 0$ and the constraint term is eliminated from the objective function. To update the condition probability for the objects, we incorporate their prior labeled information:

$$\vec{u}_{i\cdot}^t = \frac{\sum_{j=1}^{v} a_{ij} \vec{q}_{j\cdot}^t + \beta h_i \vec{f}_{i\cdot}}{\sum_{j=1}^{v} a_{ij} + \beta h_i} \quad (7)$$

In matrix forms, it would be $U^t = (D_n + \beta H_n)^{-1}(AQ^t + \beta H_n F)$ with $H_n = \text{diag}\left\{\left(\sum_{z=1}^{c} f_{iz}\right)\right\}_{n \times n}$. Note that the initial conditional probability of a labeled object is 1 at its observed class label, and 0 at all the others. However, this optimistic estimate will be changed during the updates, with the rationale that the observed labels are just random samples from some multinomial distribution. Thus we only use the observed labels to bias the updating procedure, instead of totally relying on them.

## 5 Experiments

We evaluate the proposed algorithms on eleven classification tasks from three real world applications. In each task, we have a target set on which we wish to predict class labels. Clustering algorithms are performed on this target set to obtain the grouping results. On the other hand, we learn classification models from some training sets that are in the same domain or a relevant domain with respect to the target set. These classification models are applied to the target set as well. The proposed algorithm generates a consolidated classification solution for the target set based on both classification and clustering results. We elaborate details of each application in the following.

Table 3: Classification Accuracy Comparison on a Series of Data Sets

| Methods | 20 Newsgroups | | | | | | Cora | | | | DBLP |
|---|---|---|---|---|---|---|---|---|---|---|---|
| | 1 | 2 | 3 | 4 | 5 | 6 | 1 | 2 | 3 | 4 | 1 |
| $M_1$ | 0.7967 | 0.8855 | 0.8557 | 0.8826 | 0.8765 | 0.8880 | 0.7745 | 0.8858 | 0.8671 | 0.8841 | 0.9337 |
| $M_2$ | 0.7721 | 0.8611 | 0.8134 | 0.8676 | 0.8358 | 0.8563 | 0.7797 | 0.8594 | 0.8508 | 0.8879 | 0.8766 |
| $M_3$ | 0.8056 | 0.8796 | 0.8658 | 0.8983 | 0.8716 | 0.9020 | 0.7779 | 0.8833 | 0.8646 | 0.8813 | 0.9382 |
| $M_4$ | 0.7770 | 0.8571 | 0.8149 | 0.8467 | 0.8543 | 0.8578 | 0.7476 | 0.8594 | 0.7810 | 0.9016 | 0.7949 |
| MCLA | 0.7592 | 0.8173 | 0.8253 | 0.8686 | 0.8295 | 0.8546 | 0.8703 | 0.8388 | 0.8892 | 0.8716 | 0.8953 |
| HBGF | 0.8199 | **0.9244** | 0.8811 | 0.9152 | 0.8991 | 0.9125 | 0.7834 | 0.9111 | 0.8481 | 0.8943 | 0.9357 |
| BGCM | 0.8128 | 0.9101 | 0.8608 | 0.9125 | 0.8864 | 0.9088 | 0.8687 | 0.9155 | 0.8965 | 0.9090 | 0.9417 |
| 2-L | 0.7981 | 0.9040 | 0.8511 | 0.8728 | 0.8830 | 0.8977 | 0.8066 | 0.8798 | 0.8932 | 0.8951 | 0.9054 |
| 3-L | 0.8188 | 0.9206 | 0.8820 | 0.9158 | 0.8989 | 0.9121 | 0.8557 | 0.9086 | 0.9202 | 0.9141 | 0.9332 |
| BGCM-L | **0.8316** | 0.9197 | **0.8859** | **0.9240** | **0.9016** | **0.9177** | **0.8891** | **0.9181** | **0.9246** | **0.9206** | **0.9480** |
| STD | 0.0040 | 0.0038 | 0.0037 | 0.0040 | 0.0027 | 0.0030 | 0.0096 | 0.0027 | 0.0052 | 0.0044 | 0.0020 |

**20 Newsgroup categorization.** We construct six learning tasks, each of which involves four classes. The objective is to classify newsgroup messages according to topics. We used the version [1] where the newsgroup messages are sorted by date, and separated into training and test sets. The test sets are our target sets. We learn logistic regression [15] and SVM models [7] from the training sets, and apply these models, as well as K-means and min-cut clustering algorithms [22] on the target sets.

**Cora research paper classification.** We aim at classifying a set of research papers into their areas [23]. We extract four target sets, each of which includes papers from around four areas. The training sets contain research papers that are different from those in the target sets. Both training and target sets have two views, the paper abstracts, and the paper citations. We apply logistic regression classifiers and K-means clustering algorithms on the two views of the target sets.

**DBLP data.** We retrieve around 4,000 authors from DBLP network [10], and try to predict their research areas. The training sets are drawn from a different domain, i.e., the conferences in each research field. There are also two views for both training and target sets, the publication network, and the textual content of the publications. The amount of papers an author published in the conference can be regarded as link feature, whereas the pool of titles that an author published is the text feature. Logistic regression and K-means clustering algorithms are used to derive the predictions on the target set. We manually label the target set for evaluation.

The details of each learning task are summarized in Table 2. On each target set, we apply four models $M_1$ to $M_4$, where the first two are classification models and the remaining two are clustering models. We denote the proposed method as Bipartite Graph-based Consensus Maximization (**BGCM**), which combines the output of the four models. As shown in Figure 3, only clustering ensembles, majority voting methods, and the proposed BGCM algorithm work at the meta output level where raw data are discarded and only prediction results from multiple models are available. However, majority voting can not be applied when there are clustering models because the correspondence between clusters and classes is unknown. Therefore, we compare BGCM with two clustering ensemble approaches (**MCLA** [26] and **HBGF** [12]), which ignore the label information from supervised models, regard all the base models as unsupervised clustering, and integrate the output of the base models. So they only give clustering solutions, not classification results.

To evaluate classification accuracy, we map the output of all the clustering algorithms (the base models, and the ensembles) to the best possible class predictions with the help of hungarian method [5], where cluster IDs are matched with class labels. Actually, it is "cheating" because the true class labels are used to do the mapping, and thus it should be able to generate the best accuracy from these unsupervised models. As discussed in Section 4, we can incorporate a few labeled objects, which are drawn from the same domain of the target set, into the framework and improve accuracy. This improved version of the BGCM algorithm is denoted as **BGCM-L**, and the number of labeled objects used in each task is shown in Table 2. On each task, we repeat the experiments 50 times, each of which has randomly chosen target and labeled objects, and report the average accuracy. Due to space limit, we only show the standard deviation (**STD**) for BGCM-L method. The baselines share very similar standard deviation with the reported one on each task.

**Accuracy.** In Table 3, we summarized the classification accuracy of all the baselines and the proposed approach on the target sets of eleven tasks. The two single classifiers ($M_1$ and $M_2$), and the two clustering single models ($M_3$ and $M_4$) usually have low accuracy. By combining all the base models, the clustering ensemble approaches (MCLA and HBGF) can improve the performance over each single model. However, these two methods are not designed for classification, and the reported

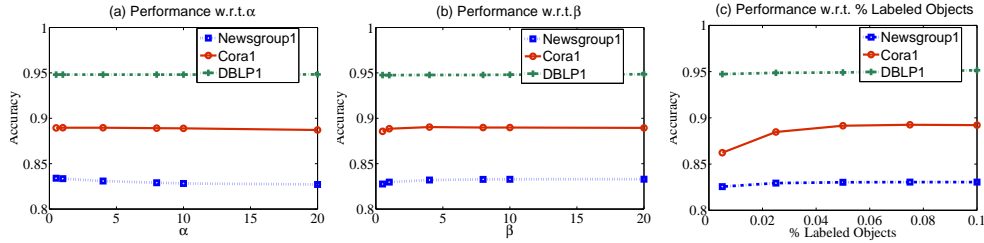

Figure 4: Sensitivity Analysis

accuracy is the upper bound of their "true" accuracy. The proposed BGCM method always outperforms the base models, and achieves better or comparable performances compared with the upper bound of the baseline ensembles. By incorporating a small portion (around 10%) of labeled objects, the BGCM-L method further improves the performances. The consistent increase in accuracy can be observed in all the tasks, where the margin between the accuracy of the best single model and that of the BGCM-L method is from 2% to 10%. Even when taking variance into consideration, the results demonstrate the power of consensus maximization in accuracy improvements.

**Sensitivity.** As shown in Figure 4 (a) and (b), the proposed BGCM-L method is not sensitive to the parameters $\alpha$ and $\beta$. To make the plots clear, we just show the performance on the first task of each application. $\alpha$ and $\beta$ are the shadow prices paid for deviating from the estimated labels of groups and observed labels of objects, so they should be greater than 0. $\alpha$ and $\beta$ represent the confidence of our belief in the labels of the groups and objects compared with 1. The labels of group nodes are obtained from supervised models and may not be correct, therefore, a smaller $\alpha$ usually achieves better performance. On the other hand, the labels of objects can be regarded as groundtruths, and thus the larger $\beta$ the better. In experiments, we find that when $\alpha$ is below 4, and $\beta$ greater than 4, good performance can be achieved. We let $\alpha = 2$ and $\beta = 8$ to get the experimental results shown in Table 3. Also, we fix the target set as 80% of all the objects, and use 1% to 20% as the labeled objects to see how the performance varies, and the results are summarized in Figure 4 (c). In general, more labeled objects would help the classification task where the improvements are more visible on Cora data set. When the percentage reaches 10%, BGCM-L's performance becomes stable.

**Number of Models.** We vary the number of base models incorporated into the consensus framework. The BGCM-L method on two models is denoted as **2-L**, where we average the performance of the combined model obtained by randomly choosing one classifier and one clustering algorithm. Similarly, the BGCM-L method on three models is denoted as **3-L**. From Table 3, we can see that BGCM-L method using all the four models outperforms the method incorporating only two or three models. When the base models are independent and each of them obtains reasonable accuracy, combining more models would benefit more because the chances of reducing independent errors increase. However, when the new model cannot provide additional information to the current pool of models, incorporating it may not improve the performance anymore. In the future, we plan to identify this upper bound through experiments with more input sources.

## 6   Conclusions

In this work, we take advantage of the complementary predictive powers of multiple supervised and unsupervised models to derive a consolidated label assignment for a set of objects jointly. We propose to summarize base model output in a group-object bipartite graph, and maximize the consensus by promoting smoothness of label assignment over the graph and consistency with the initial labeling. The problem is solved by propagating labeled information between group and object nodes through their links iteratively. The proposed method can be interpreted as conducting an embedding of object and group nodes into a new space, as well as an un-normalized personalized PageRank. When a few labeled objects are available, the proposed method uses them to guide the propagation and refine the final hypothesis. In the experiments on 20 newsgroup, Cora and DBLP data, the proposed consensus maximization method improves the best base model accuracy by 2% to 10%.

**Acknowledgement** The work was supported in part by the U.S. National Science Foundation grants IIS-08-42769, IIS-09-05215 and DMS-07-32276, and the Air Force Office of Scientific Research MURI award FA9550-08-1-0265.

## Footnotes

[1]More information, data and codes are available at http://ews.uiuc.edu/~jinggao3/nips09bgcm.htm

# References

[1] 20 Newsgroups Data Set. http://people.csail.mit.edu/jrennie/20Newsgroups/.

[2] E. Bauer and R. Kohavi. An Empirical Comparison of Voting Classification Algorithms: Bagging, Boosting, and Variants. *Machine Learning*, 36:105–139, 2004.

[3] Dimitri P. Bertsekas. Non-Linear Programming (2nd Edition). *Athena Scientific*, 1999.

[4] A. Blum and T. Mitchell. Combining Labeled and Unlabeled Data with Co-training. In *Proc. of COLT' 98*, pages 92–100, 1998.

[5] N. Borlin. Implementation of Hungarian Method. http://www.cs.umu.se/∼niclas/matlab/assignprob/.

[6] R. Caruana. Multitask Learning. *Machine Learning*, 28:41–75, 1997.

[7] C.-C. Chang and C.-J. Lin. LibSVM: a Library for Support Vector Machines, 2001. Software available at http://www.csie.ntu.edu.tw/∼cjlin/libsvm.

[8] O. Chapelle, B. Schölkopf and A. Zien (eds). Semi-Supervised Learning. *MIT Press*, 2006.

[9] K. Crammer, M. Kearns and J. Wortman. Learning from Multiple Sources. *Journal of Machine Learning Research*, 9:1757-1774 , 2008.

[10] DBLP Bibliography. http://www.informatik.uni-trier.de/∼ley/db/.

[11] T. Dietterich. Ensemble Methods in Machine Learning. In *Proc. of MCS '00*, pages 1–15, 2000.

[12] X. Z. Fern and C. E. Brodley. Solving Cluster Ensemble Problems by Bipartite Graph Partitioning. In *Proc. of ICML' 04*, pages 281–288, 2004.

[13] K. Ganchev, J. Graca, J. Blitzer, and B. Taskar. Multi-view Learning over Structured and Non-identical Outputs. In *Proc. of UAI' 08*, pages 204–211, 2008.

[14] J. Gao, W. Fan, Y. Sun, and J. Han. Heterogeneous source consensus learning via decision propagation and negotiation. In *Proc. of KDD' 09*, pages 339–347, 2009.

[15] A. Genkin, D. D. Lewis, and D. Madigan. BBR: Bayesian Logistic Regression Software. http://stat.rutgers.edu/∼madigan/BBR/.

[16] A. Goldberg and X. Zhu. Seeing stars when there aren't many stars: Graph-based semi-supervised learning for sentiment categorization. In *HLT-NAACL 2006 Workshop on Textgraphs*.

[17] A. Gionis, H. Mannila, and P. Tsaparas. Clustering Aggregation. *ACM Transactions on Knowledge Discovery from Data*, 1(1), 2007.

[18] T. Haveliwala. Topic-Sensitive PageRank: A Context-Sensitive Ranking Algorithm for Web Search. *IEEE Transactions on Knowledge and Data Engineering*, 15(4):1041-4347, 2003.

[19] J. Hoeting, D. Madigan, A. Raftery, and C. Volinsky. Bayesian Model Averaging: a Tutorial. *Statistical Science*, 14:382–417, 1999.

[20] R. Jacobs, M. Jordan, S. Nowlan, and G. Hinton. Adaptive Mixtures of Local Experts. *Neural Computation*, 3:79-87, 1991.

[21] T. Joachims. Transductive Learning via Spectral Graph Partitioning. In *Proc. of ICML' 03*, pages 290–297, 2003.

[22] G. Karypis. CLUTO – Family of Data Clustering Software Tools. http://glaros.dtc.umn.edu/gkhome/views/cluto.

[23] A. McCallum, K. Nigam, J. Rennie, and K. Seymore. Automating the Construction of Internet Portals with Machine Learning. *Information Retrieval Journal*, 3:127–163, 2000.

[24] L. Page, S. Brin, R. Motwani, and T. Winograd. The PageRank Citation Ranking: Bringing Order to the Web. *Technical Report, Stanford InfoLab*, 1999.

[25] V. Singh, L. Mukherjee, J. Peng, and J. Xu. Ensemble Clustering using Semidefinite Programming. In *Proc. of NIPS' 07*, 2007.

[26] A. Strehl and J. Ghosh. Cluster Ensembles – a Knowledge Reuse Framework for Combining Multiple Partitions. *Journal of Machine Learning Research*, 3:583–617, 2003.

[27] D. Wolpert. Stacked Generalization. *Neural Networks*, 5:241–259, 1992.

[28] D. Zhou , J. Weston, A. Gretton, O. Bousquet and B. Scholkopf. Ranking on Data Manifolds. In *Proc. of NIPS' 03*, pages 169–176, 2003.

[29] X. Zhu. Semi-supervised Learning Literature Survey. Technical Report 1530, Computer Sciences, University of Wisconsin-Madison, 2005.

